# When to Act and When to Ask: Policy Learning With Deferral Under Hidden Confounding

**Marah Ghoummaid**
Faculty of Data and Decision Sciences, Technion
marahghoummaid@gmail.com

**Uri Shalit**
Faculty of Data and Decision Sciences, Technion
urishalit@tauex.tau.ac.il

## Abstract

We consider the task of learning how to act in collaboration with a human expert based on observational data. The task is motivated by high-stake scenarios such as healthcare and welfare, where algorithmic action recommendations are made to a human expert, opening the option of deferring recommendation in cases where the human might act better on their own. This task is especially challenging when dealing with observational data, as using such data runs the risk of hidden confounders whose existence can lead to biased and harmful policies. However, unlike standard policy learning, the presence of a human expert can mitigate some of these risks. We build on the work of Mozannar and Sontag [2020] on consistent surrogate loss for learning with the option of deferral to an expert, where they solve a cost-sensitive supervised classification problem. Since we are solving a causal problem, where labels do not exist, we use a causal model to learn costs which are robust to a bounded degree of hidden confounding. We prove that our approach can take advantage of the strengths of both the model and the expert to obtain a better policy than either. We demonstrate our results by conducting experiments on synthetic and semi-synthetic data and show the advantages of our method compared to baselines.

## 1  Introduction

Machine learning models are increasingly being developed to perform tasks performed by human decision-makers in high-stakes settings such as clinical decision making [Adams et al., 2022, Rajpurkar et al., 2022], criminal justice [Stevenson and Doleac, 2022] and social services [Behncke et al., 2009, McBrien et al., 2022]. Some of these tasks involve recommending actions such as medical treatment, releasing on bail, or receiving benefits. Typically the data used to train such action-recommendation models is based on past human decisions and their outcomes. For example, we might observe how patients were treated for diabetes and their subsequent health outcomes.

Learning to act better in the future based on past observed actions is a *causal* problem. For such problems we face the fundamental problem of causal inference [Holland, 1986], i.e. the unknowability of counterfactual outcomes: "Would this patient have been better had they been treated differently?". Estimating causal quantities based on observational data such as hospital records is risky, as there is always the possibility of hidden confounding. Roughly speaking, this means there exist factors that affected the human decision maker and the outcome, but are unavailable to the model during training.

Learning models from data which has hidden confounding can lead to biased and harmful treatment assignment policies. However, human experts' decisions can also be biased, sub-optimal, or wrong. Thus, a setting in which both the human expert and the model can complement each other might be the best choice, mitigating the potential weaknesses and leveraging the strengths of both the model and the human expert [Bansal et al., 2020, Charusaie et al., 2022]. We believe this is especially pertinent in causal inference, as the human expert typically has access to the hidden confounders.

In this paper, we design a framework for learning a causal action recommendation model that can jointly work with a human decision-maker, while using observational data where hidden confounders might exist. We assume that the effect of the hidden confounders is limited (in a way we specify below), as without any assumption about the nature of hidden confounding, learning would be impossible. Our goal is to design a system consisting of a machine learning model estimating causal effects and a human expert working in a complementary setting. In addition to learning treatment assignment, the model learns the weaknesses and strengths of the human expert and will decide in each case whether to recommend a certain treatment to the patient, or whether it is better to defer the decision to the expert. The ultimate goal is to learn better policies and to reduce the burden on human experts. We call our method Causal Action Recommendation with Expert Deferral (CARED).

CARED follows in the footsteps of Mozannar and Sontag [2020], who developed a method for learning a classifier with the option of deferral to an expert. They propose a reduction of the problem to a cost-sensitive learning problem, where costs are based on the true labels of data samples. They give a surrogate loss for the cost-sensitive learning problem which generalizes the cross-entropy loss, and prove this loss is consistent, i.e., converges to the optimal solution of the original problem. Our problem can similarly be viewed as a classification problem with the option of deferring to an expert. However, the causal case is more difficult: the correct label for each sample is the best treatment to be prescribed to this sample based on its features, and due to the fundamental problem of causal inference we cannot know this label. Thus, unlike Mozannar and Sontag [2020] we do not have the true labels for our classification problem.

To overcome the challenges of the causal setting, we propose a set of costs that instead of being based on the true labels, are based on estimated *bounds* on counterfactual outcomes. These costs guide our model towards learning what is the right treatment for each patient while acknowledging that the human expert has access to additional information the model cannot access. We prove a generalization bound on the loss of the joint machine-expert system, and further prove that under certain assumptions about the model used to estimate the bounds mentioned above, the joint machine-expert system outperforms both the human expert and a pure machine learning model. Finally, we evaluate CARED on synthetic and semi-synthetic data to demonstrate how we can learn policies that outperform both pure machine learning policies and human experts. We further show that CARED outperforms a recently proposed method by Gao and Yin [2023] that addresses the same problem with an inverse-propensity weighted approach.

## 2 Related work

Many works focus on solving the problem of policy learning for action recommendations from observational data. Some notable approaches include reweighting by inverse propensity weighting (IPW) and other weighting techniques [Swaminathan and Joachims, 2015, Kallus, 2017, Beygelzimer and Langford, 2009], and the approach of using doubly robust scores to determine the optimal treatment assignment policy for binary treatments [Dudík et al., 2014, Athey and Wager, 2021,?, Kallus and Zhou, 2020]. Other methods predict the Conditional Average Treatment Effect (CATE) and use it as the guideline for treatment assignment for each sample, such as Jesson et al. [2021], and Kallus et al. [2019]. Most of these works assume ignorability, i.e., that there are no hidden confounders that affect both treatment assignment and the outcome in the data. As mentioned earlier, this assumption rarely holds when observational data is in use, and its presence, if not accounted for, can lead to biased and harmful policies. Our work builds on previous work for learning supervised classification problems with the deferral option by Mozannar and Sontag [2020], and is inspired by Athey and Wager [2021] who derive costs for learning policies from observational data under the assumption of no hidden confounders.

Gao and Yin [2023] present a framework for collaborative human-AI policy learning from observational data with deferral, building on earlier work [Gao et al., 2021] which did not allows for hidden confounding. To the best of our knowledge, theirs is the only existing method that learns a policy with the deferral option under hidden confounding. Their method minimizes an inverse propensity weighted estimator of the worst-case risk over a class of differentiable policies, and over an uncertainty set around the observed propensities. The uncertainty set is determined by constraints motivated by the Marginal Sensitivity Model [Tan, 2006]. While our method employs both outcome models and propensity scores, Gao and Yin [2023]'s approach focuses on propensity score re-weighting. The re-weighted objective implies that only cases where the proposed policy agrees to a high degree with the observed policy are taken into account. As we show in the experimental section

below, this (along with the lack of outcome model) might lead to under-performance, especially in the (common) case where the constraint set is not accurately known. Gao and Yin [2023] also explore the case where there are multiple specific human experts the model is optimizing for, which we plan to explore in future work. We also compare our method to Kallus and Zhou [2020], who take a similar approach as Gao and Yin [2023] for learning a policy while allowing the violation of the unconfoundedness assumption. However, they do not have the option of deferral.

In other related work, Stensrud et al. [2024] consider the case where the expert's action can be used as input to the method, motivated by the fact that the expert typically has access to unobserved confounders. This is distinct from our use case but has interesting implications in identifying the so-called "superoptimal regime" where the expert's action can strictly improve over a policy derived purely from the observables. Finally, Yin et al. [2024] offer a novel approach towards learning to defer in the non-causal setting, which could be adapted to our use case in the future.

## 3 Setup

We work under an observational data setting with the Neyman-Rubin potential outcomes framework [Rubin, 2005]. Let $(X, A, Y(1), Y(0), U)$ be a sample drawn from the unobservable distribution $P_{\text{full}}$, where $A \in \mathcal{A} = \{0, 1\}$ is a binary treatment, $X \in \mathcal{X} \subset \mathbb{R}^d$ is a set of baseline covariates, $Y(1)$ and $Y(0)$ are the real-valued treated and untreated potential outcomes, respectively, and $U \in \mathbb{R}^k$ is an unobserved confounder. We face the fundamental problem of causal inference and only observe $n$ draws from the coarsened distribution $P$ over the observed variables $Z = (X, A, Y)$, where we assume that $Y = Y(A)$, i.e. (causal) consistency. We generally follow the convention that higher outcomes are better, we mention when assumed differently. We use the Marginal sensitivity Model (MSM) [Tan, 2006] as a way to model a limited degree of unobserved confounding. We are interested in learning a policy with the option of deferral, such that given the patient's covariates it either assigns a treatment or defers the decision to an expert.

Let $e(x) = P(A = 1 \mid X = x)$ and $e(x, u) = P_{\text{full}}(A = 1 \mid X = x, U = u)$ be the observed and full propensity scores, under $P$ and $P_{\text{full}}$, the observed and the full unobserved distributions, respectively.

**Assumption 1** (MSM Assumption). We assume $e(x), e(x, u) \in (0, 1)$ and that the ratio between the full odds of treatment $e(x, u)/(1 - e(x, u))$ and the observed odds of treatment $e(x)/(1 - e(x))$ is bounded by at most a factor of $\Lambda \geq 1$ almost surely under $P_{\text{full}}$:

$$\Lambda^{-1} \leq \frac{e(x, u)}{1 - e(x, u)} \Big/ \frac{e(x)}{1 - e(x)} \leq \Lambda.$$

Note that When $\Lambda = 1$, Assumption 1 is equivalent to the classic assumption of unconfoundedness with respect to the observed $X$. As $\Lambda$ increases, the MSM allows for greater levels of unobserved confounding. Setting $\Lambda$ is a matter of ongoing research, and is typically done by calibrating with respect to observed confounders [McClean et al., 2024]. In our case, it is also tied to the fraction of deferrals to the human expert and can be tuned to achieve a desired deferral rate.

For a treatment $a \in \{0, 1\}$ and a covariate vector $x \in \mathcal{X}$ we define the **Conditional Average Potential Outcome (CAPO)** as $\mathbb{E}[Y(a)|X = x]$. The **Conditional Average Treatment Effect (CATE)** is the difference $\tau(x) = \mathbb{E}[Y(1) - Y(0)|X = x]$. Let $\mathcal{M}(\Lambda)$ be the set of distributions $\tilde{P}_{\text{full}}$ that agree with the observed $P$ on $(X, A, Y)$ and that also agree with Assumption 1. Then $Y^+(x, a) = \sup_{\tilde{P}_{\text{full}} \in \mathcal{M}(\Lambda)} \mathbb{E}[Y(a)|X = x]$ is the so-called sharp upper bound on the CAPO $\mathbb{E}[Y(a)|X = x]$, and similarly for $Y^-(x, a)$, taking an infimum instead. We further let $\hat{Y}^+(x, a), \hat{Y}^-(x, a)$ be estimated upper and lower bound for the corresponding CAPO functions.

A **policy** with deferral is a function $\pi : \mathcal{X} \to \{0, 1, \perp\}$ that maps covariates to a possible action, or defers the decision to an expert, where $\perp$ denotes deferral. A policy $\pi$ can be assessed by its **policy value** $V(\pi) = \mathbb{E}[Y(\pi)]$, where higher policy values are related to better policies unless stated otherwise. The action when the algorithm chooses to defer is the action the human expert has taken, see discussion of this in the limitations subsection (8). The learned policies will be assessed relative to a baseline policy value, such as the policy value based solely on the human expert. Our goal is for CARED to learn a policy that does no worse than the baseline policy and hopefully outperforms it.

# 4 CAPO-Based Policies

We now present two baseline policies that can be defined based on the CAPO bounds.

The first is **Bounds Policy** Equation (1). This policy assigns treatment if the upper and lower bounds on the CATE have the same sign, and otherwise it defers, i.e. it defers if the CATE interval crosses 0. This approach was used in previous work, e.g. [Jesson et al., 2021, Oprescu et al., 2023]. The second is the **Pessimistic Policy** Equation (2). It is pessimistic in the sense that it does not trust the human expert, so it does not allow deferral to an expert. It agrees with the Bounds Policy in the cases where that policy would not defer, and in the cases where the Bounds Policy would defer it decides based on the lower bounds, which (assuming higher outcomes are better) are indeed pessimistic.

Let $\hat{Q}(x) = (\hat{Y}^+(x,0), \hat{Y}^-(x,0), \hat{Y}^+(x,1)), \hat{Y}^-(x,1))$ be the CAPO bounds estimates for a sample $(X = x, A = a, Y = y)$. We further assume that the expert policy is reflected in the training data in the sense that their policy is $\pi_{\exp}(x_i) = a_i$. Then, given a model that supplies CAPO bounds $\hat{Q}$, we define the following policies, which we consider as baselines henceforth:

**Bounds Policy:**

$$\pi_{\text{bounds}}^{\hat{Q}}(x) = \begin{cases} 1 & \text{if } \hat{Y}^-(x,1) - \hat{Y}^+(x,0) > 0 \\ 0 & \text{if } \hat{Y}^+(x,1) - \hat{Y}^-(x,0) < 0 \\ \perp & \text{otherwise.} \end{cases} \tag{1}$$

**Pessimistic Policy:**

$$\pi_{\text{pessimistic}}^{\hat{Q}}(x) = \begin{cases} 1 & \text{if } \hat{Y}^-(x,1) - \hat{Y}^+(x,0) > 0 \\ 0 & \text{if } \hat{Y}^+(x,1) - \hat{Y}^-(x,0) < 0 \\ 1 & \text{otherwise}, \hat{Y}^-(x,1) - \hat{Y}^-(x,0) > 0 \\ 0 & \text{otherwise} \end{cases} \tag{2}$$

The above policies, while accounting for hidden confounding, even when allowing deferral to an expert, do not learn the strengths and weaknesses of the expert, so they might not be the optimal policies. We show this theoretically in Section 6.2, and empirically in Section 7.

# 5 Method

In this section, we present our method in which we design a machine-expert system for learning a policy with the option of deferral to an expert under hidden confounding. In 5.1 we describe the joint machine-expert objective we are interested in optimizing, the challenges it imposes, and how to optimize a consistent surrogate cost-sensitive loss that converges to the optimal solution of the original objective, based on Mozannar and Sontag [2020]. Then in 5.2 we introduce our proposed set of costs based on the CAPO bounds for solving this cost-sensitive problem. Lastly, we explain the step of CAPO estimation in 5.3. Our algorithm is summarized below in Algorithm 1.

## 5.1 Joint machine-expert objective function

We design a joint machine-expert system, where we aim to learn a policy $\pi : \mathcal{X} \to \mathcal{A} \cup \{\perp\}$. We denote by $m \in \mathcal{A}$ the expert's action that is assumed to be drawn from the distribution $M|X = x, U = u$. Note that the expert might have access to additional information – the hidden confounder $U$, which is unavailable to the model. We cast our problem as a cost-sensitive optimization objective

$$L(\pi) = \mathbb{E}_{x,y \sim P(X,Y), m \sim M|x,u}[C(x, \pi(x))\mathbb{I}_{\pi(x) \neq \perp} + C_\perp(x, m, y)\mathbb{I}_{\pi(x) = \perp}], \tag{3}$$

where $C(x, a)$ is the cost incurred by the system if for a sample with covariates $x$, an action/treatment $a$ was chosen by the model, and $C_\perp(x, m, y)$ is the cost incurred when action $m$ was chosen for sample $x$ by the expert. We explain below why $C_\perp$ depends on $y$ but $C$ does not.

In the case of no unobserved confounding and no deferral, Athey and Wager [2021] have shown how the costs above can be set such that the cost minimizer is the policy with optimal policy value. We describe how we set costs that address both hidden confounding and deferral in the next subsection.

The above objective is non-convex and difficult to optimize. We deal with this challenge by building on the approach of Mozannar and Sontag [2020] for learning classifiers with the option of deferral to

an expert, where they have a similar objective as in Equation (3). They give a convex and consistent surrogate loss for the cost-sensitive learning problem, which is a weighted cross-entropy loss, where the weights are based on a set of costs they build using the true labels for the classification problem.

As explained earlier, our problem can also be viewed as a classification problem with the option of deferring to an expert. However, due to the fundamental problem of causal inference [Holland, 1986] we do not have the true labels, which are in our case, the best treatment to be prescribed to this sample based on its features, which is a challenge that we deal with in Section 5.2.

Following is the surrogate loss we optimize, in Section 5.2 we present our proposed set of costs, and in Section 6 we give consistency and generalization guarantees for this surrogate loss.

Let $\pi_i : \mathcal{X} \rightarrow \mathbb{R}$ be the raw output of the policy $\pi$ corresponding to a class $i \in \{0, 1, \perp\}$, and define $\pi(x) = \arg\max_{i \in \{0,1,\perp\}} \pi_i(x)$. Let $z = (x, a, y)$ be a sample, and $\hat{Q}(x) = (\hat{Y}^+(x, 0), \hat{Y}^-(x, 0), \hat{Y}^+(x, 1)), \hat{Y}^-(x, 1))$ the CAPO bounds. These bounds are then used to define the scores $c(0) = C(x, 0)$, $c(1) = C(x, 1)$, and $c(\perp) = C_\perp(x, m, y)$ as we show in Section 5.2 below. Define $w^j(z, \hat{Q}(x)) = \max_{k \in \{0,1,\perp\}} c(k) - c(j)$. Then the surrogate loss function for Equation (3) is given by:

$$L_{CE}(\pi, z; \hat{Q}) = \sum_{j \in \{0,1,\perp\}} -w^j(z, \hat{Q}(x)) \log \left( \frac{\exp(\pi_j(x))}{\sum_{k \in \{0,1,\perp\}} \exp(\pi_k(x))} \right). \quad (4)$$

The method we propose uses the above system loss for learning a policy from observational data with the presence of a limited degree of hidden confounding and the ability to defer. The difficulty in this case is constructing the costs $C(x, 0), C(x, 1)$ and $C_\perp(x, m, y)$ which are used to derive the weights $w^j$, since ground truth labels are never available — a challenge we address now.

## 5.2 Action Costs

As mentioned earlier, our classification problem does not have the true labels, as opposed to Mozannar and Sontag [2020]. We therefore propose a set of costs based on estimated bounds on counterfactual outcomes rather than on the true labels, guiding our model in learning for which cases it can safely recommend actions, and which to defer to the human expert. These costs are then used to derive the weights in the objective Equation (4).

For $i \in [n]$ let $(x_i, y_i, a_i)$ be the $i$-th observed sample, and let $\hat{Y}^+(x_i, a), \hat{Y}^-(x_i, a)$ be the estimates of the upper and lower bounds of the CAPO for an action $a \in \mathcal{A}$, and a covariates $x \in \mathcal{X}$. We propose setting the cost of assigning the action $a$ to the $i$-th sample to be:

$$C(x_i, a) = \hat{Y}^+(x_i, 1 - a) - \hat{Y}^-(x_i, a).$$

This cost encourages the model to choose the action with the highest outcome. When this is not the case, i.e. the model chooses another action than the one with the highest estimated outcome, then it will incur a cost which is the difference between the *lower bound* of the outcome corresponding to the action chosen, and the *upper bound* of the outcome corresponding to the action which was supposed to be chosen based on the outcomes. This cost can be thought of as the *worst-case* regret of the model relative to the optimal action. It is important to note that this is only one option among many options for building a set of costs for this purpose. For instance, we can switch the roles of the upper and lower bounds of the CAPOs to obtain various costs expressing differing levels of risk-aversion.

As for the cost of deferring to an expert, following the same logic, we propose two alternatives. Given covariates $x_i$, expert action $m = a_i$ and outcome $y_i$ the *conservative* deferral cost is:

$$C_\perp(x_i, a_i, y_i) = \hat{Y}^-(x_i, 1 - a_i) - y_i,$$

and similarly the *optimistic* deferral cost is: $C_\perp(x_i, a_i, y_i) = \hat{Y}^+(x_i, 1 - a_i) - y_i$. These costs represent to what extent are we willing to take an action different from the one the expert took which resulted in the outcome actually seen in the data, i.e. $m = a_i$ which lead to outcome $y_i$. In the optimistic case, deferring to an expert results in incurring a greater cost in comparison to its conservative counterpart, encouraging the model to make fewer deferrals and focusing on samples it is highly uncertain about. It is optimistic in the sense of assuming the model's estimates are likely

correct. Choosing between these alternatives should to be done based on the characteristics of the specific policy and use case. In Appendix A.3 we show a comparative analysis of the two alternatives.

To summarize we present here the costs corresponding to the *conservative* approach:

$$C(x_i, 1) = \hat{Y}^+(x_i, 0) - \hat{Y}^-(x_i, 1)$$

$$C(x_i, 0) = \hat{Y}^+(x_i, 1) - \hat{Y}^-(x_i, 0)$$

$$C_\perp(x_i, a_i, y_i) = \begin{cases} \hat{Y}^-(x_i, 0) - y_i, & \text{if } a_i = 1 \\ \hat{Y}^-(x_i, 1) - y_i, & \text{otherwise.} \end{cases}$$

Taken together, these costs encourage the model to classify a sample as $a = 1$ or $a = 0$ in cases where it is certain that particular action would be best, and to defer cases where the expert seems to have made the correct decision (in the conservative case). An illustrative example of the costs choice can be found in Appendix A. Additionally, we demonstrate costs' coherence in Theorem 1.

## 5.3 CAPO Estimation

From an algorithmic point of view, any method that yields upper and lower bounds on the CAPO given some degree of hidden confounding can be used to obtain the costs Section 5.2. As we will see in the next section, our theoretical results require bounds with certain generalization and validity properties to hold. In this work, we use the B-learner [Oprescu et al., 2023] for estimating the upper and lower bounds of the CAPO, as it has the properties needed for the theoretical analysis and shows good performance in practice, and being a meta-learner it can accommodate various base learners, including random forests and neural networks.

---

**Algorithm 1** CARED Policy Learner - The Conservative Approach

---

**input** Data $\{(x_i, a_i, y_i) : i \in \{1, ..., n\}\}$
 1: Use data $\{(x_i, a_i, y_i) : i \in \{1, ..., n\}\}$ to learn the CAPO bounds:
   $\hat{Q}(x_i) = (\hat{Y}^+(x_i, 0), \hat{Y}^-(x_i, 0), \hat{Y}^+(x_i, 1)), \hat{Y}^-(x_i, 1))$
   `//e.g. using B-learner` [Oprescu et al., 2023]
 2: Construct the *conservative* costs based on the CAPO bounds:

$$c_i(1) = C(x_i, 1) = \hat{Y}^+(x_i, 0) - \hat{Y}^-(x_i, 1)$$

$$c_i(0) = C(x_i, 0) = \hat{Y}^+(x_i, 1) - \hat{Y}^-(x_i, 0)$$

$$c_i(\perp) = C_\perp(x_i, a_i, y_i) = \begin{cases} \hat{Y}^-(x_i, 0) - y_i, & \text{if } a_i = 1 \\ \hat{Y}^-(x_i, 1) - y_i, & \text{otherwise} \end{cases}$$

 3: Calculate weights $w_i^j = \max_{k \in \{0,1,\perp\}} c_i(k) - c_i(j)$
 4: Learn a policy: $\pi \in \arg\min_{\pi' \in \Pi} \frac{1}{n} \sum_{i=1}^n \sum_{j \in \{0,1,\perp\}} -w_i^j \log \left( \frac{\exp(\pi'_j(x_i))}{\sum_{k \in \{0,1,\perp\}} \exp(\pi'_k(x_i))} \right)$.
**output** $\pi$

---

## 6 Theoretical Guarantees

We present theoretical guarantees for CARED as follows: In 6.1 we show that the optimum of the surrogate loss function $L_{CE}$ (eq. 4) agrees with the optimum of the machine-expert loss function $L$ (eq. 3). Then, in 6.2 we show that the costs we use in $L_{CE}$ are coherent, in the sense that minimizing them leads to a decision that is non-inferior to the decision either the expert or the machine would have made on their own. Finally, in 6.3 we give a generalization bound for the loss $L_{CE}$.

### 6.1 Consistency

**Corollary 1.** $L_{CE}$ *is convex in $\pi$ and is a consistent loss function for $L$:*
*Let $\tilde{\pi} = \arg\inf_\pi \mathbb{E}[L_{CE}(\pi, z; \hat{Q})]$, then $\tilde{\pi} = \arg\inf_\pi L(\pi)$.*

This result is a straightforward adaptation of Proposition 1 of Mozannar and Sontag [2020], as their result is not sensitive to the particular choice of costs (B.0.1). It motivates using the surrogate loss function $L_{CE}$ which is much more amenable to optimization than the original machine-expert loss $L$.

## 6.2 Costs Are Coherent

We now show that the costs $C(x, 0), C(x, 1), C_\perp(x, a, y)$ are coherent, in the sense that they indeed work as intended: a policy that minimizes them locally is always at least as good as (and indeed, in general better than) both the expert on their own and the machine on its own.

**Definition 1.** [Bound Validity] For a sample $x$ with corresponding potential outcomes $Y(0)$ and $Y(1)$, let $\hat{Q}(x) = (\hat{Y}^+(x, 0), \hat{Y}^-(x, 0), \hat{Y}^+(x, 1)), \hat{Y}^-(x, 1))$ be the estimated CAPO bounds. Then $\hat{Q}$ is valid for $(Y(0), Y(1))$ if $Y(a) \in [\hat{Y}^-(x, a), \hat{Y}^+(x, a)]$ for $a \in \{0, 1\}$.

Validity means that bounds indeed contain their respective potential outcomes. For example, Oprescu et al. [2023] prove that their bounds are valid on average.

**Theorem 1** (Costs are coherent). *Let $\tilde{\pi}(x_i) \in \arg\min L_{CE}(\pi, z_i; \hat{Q}(x_i))$, $\pi_{exp}$ the expert's policy, and $\pi_{bounds}^{\hat{Q}}(x_i)$ the CAPO-based policy defined in 1. If $\hat{Q}(x_i)$ is valid for $(Y_i(0), Y_i(1))$, then*

$$Y(\tilde{\pi}(x_i)) \geq \max \left\{ Y\left(\pi_{exp}(x_i)\right), Y\left(\pi_{bounds}^{\hat{Q}}(x_i)\right) \right\}.$$

*Furthermore, under certain technical conditions on the distribution of $Y(0), Y(1)$ and $\hat{Q}$, the inequality is strong with non-zero probability for each sample.*

Theorem 1 shows that whenever the bounds include the true potential outcome, the action that minimizes our proposed loss function is at least as good as the action implied by the baseline $\pi_{bounds}^{\hat{Q}}$ policy, as well as the human expert policy.

## 6.3 Generalization Bound

We now show the generalization bound for our machine-expert system loss. We start with assumptions, then we state the main theorem.

**Assumption 2** (Policy learners). Let $\Pi$ be the class of policies over which we optimize Equation (4). We assume the class $\Pi$ is with restricted complexity, specifically $\mathcal{R}_n(\Pi) = O(\frac{1}{\sqrt{n}})$, where $\mathcal{R}_n(\Pi)$ is the Rademacher Complexity of the policy class $\Pi$. Classes that have this property include linear functions, logistic functions, decision trees with a bounded depth, and neural networks with weight decay or dropout Kallus and Zhou [2020].

**Assumption 3** (Policy Learner with Bounded Outputs). Let $\pi \in \Pi$ and $\pi_j : \mathcal{X} \to \mathbb{R}$ be the raw output of the classifier $\pi$ corresponding to a class/action $j \in \{0, 1, \perp\}$. We assume there exists a constant $C_\pi$ such that $|\pi_j| \leq C_\pi$ for all $j$.

**Assumption 4** (Boundedness of the Outcomes). $Y$ is bounded, i.e. $|Y| \leq C_Y$ for $C_Y > 0$.

**Assumption 5.** [Rates for ERM CAPO Bounds Estimators] The CAPO bounds estimators $\hat{Y}^\diamond(x, a)$ for $a \in \{0, 1\}$ and $\diamond \in \{+, -\}$ satisfy: $\|\hat{Y}^\diamond(x, a) - Y^\diamond(x, a)\| \lesssim O_p(n^{-1/(2+r)})$ with $0 < r < 2$.

Assumption 5 implies that the convergence rate of the $L_2$ norm of the estimation error of the CAPO bounds converges not too slowly (in probability).Notably, the B-Learner bound estimates satisfy Assumption 5 under some assumption on the class of policy learners and the nuisance estimators, according to Corollary 1 from Oprescu et al. [2023].

We now give a generalization bound on the loss of the joint machine-expert system:

**Theorem 2** (Generalization Bound). *Given a policy class $\Pi$ satisfying Assumption 2, Assumption 3 with a constant $C_\pi$, Assumption 4 with a constant $C_Y$, and Assumption 5 with $0 < r < 2$.*

*Let $\pi \in \Pi$, be a policy, and let $Q, \hat{Q}$ be the CAPO bounds and the estimated CAPO bounds respectively. Then there exists a constant $C > 0$ such that with a probability of at least $1 - \delta$ the following holds:*

$$L_D(\pi; Q) - L_S(\pi; \hat{Q}) \leq 2\mathcal{R}_n(\Pi) + 24 \cdot C_\pi \cdot \left( C_Y \sqrt{\frac{2\ln(4/\delta)}{n}} + C \cdot \left(n^{-1/(2+r)}\right) \right) \quad (5)$$

*where $L_S(\pi; \hat{Q}) = \frac{1}{n}\sum_{i=1}^n L_{CE}(\pi, z_i; \hat{Q}(x_i))$ is the training loss, and $L_D(\pi; Q) = \mathbb{E}_{z \sim p(z)}[L_{CE}(\pi, z; Q(x))]$ is the expected loss for the combined machine-expert system loss in 4, and $\mathcal{R}_n(\Pi)$ is the Rademacher Complexity of the policy class $\Pi$.*

# 7 Experiments

Here we examine the utility of CARED by conducting experiments on synthetic and semi-synthetic data in Section 7.1 and Section 7.2, respectively. Further details about the experiments, datasets, models, and hyper-parameters can be found in Appendix C.

Our main comparison is to Gao and Yin [2023]'s method **(ConfHAI)** that allows for a bounded degree of confounding in addition to allowing the option of deferral to a human expert. They learn a policy by optimizing a minimax reweighting-based risk estimate over an uncertainty set around the observed propensities. The uncertainty sets are determined by the MSM assumption, which is the same assumption we employ to bound the degree of hidden confounding. Another baseline we consider is Kallus and Zhou [2020]'s method **(CRLogit)** which proceeds under a similar approach and allows for a bounded degree of confounding, but does not allow deferral to a human. Additionally, we evaluate all methods against the following baselines: **Oracle Policy** which is the best policy that assigns for each patient the true best treatment. In both experiments, this policy is available to us, as we have the true potential outcomes. **Current Expert/ Baseline Policy** this is the default policy that we refer to in cases of deferral. Typically, it is the policy of the current expert, but it can be any other policy of our choice. **Pessimistic Policy**: Using the Equation (2) policy with the bounds $\hat{Q}$ given by the B-learner Oprescu et al. [2023]. **B-Learner Policy**: Using the $\pi_{\text{bounds}}^{\hat{Q}}$ policy of Equation (1) with the bounds $\hat{Q}$ given by the B-learner [Oprescu et al., 2023]. We use the exact same $\hat{Q}$ in Algorithm 1, and in the B-learner and pessimistic policies. **Random Deferral Policy**: this is a variant of the B-Learner policy which introduces a different approach to deferral, where samples are deferred randomly based on a specified deferral rate.

## 7.1 Synthetic data

In this experiment, we replicate the study from Gao and Yin [2023] using synthetic data, comparing our method to **ConfHAI** and **CRLogit**. Since this experiment is measuting regret, we follow Gao et al. [2021] and compare to a **Baseline Policy** that assigns treatment $a = 0$ for all patients and is denoted by $\pi_0$. In this experiment lower outcomes are better.

**Data** The data is generated according to the following data generation process:

$$\xi \sim \text{Bern}(0.5), \quad X \sim \mathcal{N}((2\xi - 1)\mu_x, I_5),$$
$$U = \mathbb{I}[Y(1) < Y(0)],$$
$$Y(A) = \beta_0^\top x + \mathbb{I}[A = 1]\beta_{treat}^\top x + 0.5\alpha\xi\mathbb{I}[A = 1] + \eta + \omega\xi + \epsilon$$

where $\beta_0 = [0, 0.5, -0.5, 0, 0]$, $\beta_{treat} = [-1.5, 1, -1.5, 1, 0.5]$, $\mu_x = [-1, 0.5, -1, 0, -1]$, $\eta = 2.5$, $\alpha = -2$, $\omega = 1.5$, and $\epsilon \sim \mathcal{N}(0, 1)$. The nominal propensity is logistic by $X$, $e(X) = \sigma(\beta^\top X)$ with $\beta = [0, 0.75, -0.5, 0, -1, 0]$. The confounder is denoted by $U$, and the true propensity score is given by: $e(X, U) = \frac{(\Lambda_0 U + 1 - U)e(X)}{[1 + 2(\Lambda_0 - 1)e(X) - \Lambda_0]U + \Lambda_0 + (1 - \Lambda_0)e(X)}$, with the true $\Lambda_0$, such that $log(\Lambda_0) = 2.5$.

**Experiment** We replicate the experiment from Gao and Yin [2023] and run 10 trials, with different instances of the above data, a train data size of 2000, and test data of size 10000. For each trial, we vary the sensitivity parameter $\Lambda$ in $\{0.01, 0.5, 1, 1.5, 2, 2.5, 3, 3.5, 4\}$, corresponding to various levels of assumed hidden confounding. We compare the *policy regret* for the returned policy for each method relative to the **Baseline Policy**. In addition, we report the policy regret for the human expert in the dataset **(Human's Policy)** which is reflected by the variable $A$ in the dataset. As for the CARED policy, we obtain it by applying Algorithm 1 with a logistic policy implemented as a single-layer MLP network.

**Results** In Figure 1 we see that CARED outperforms all other methods, specifically improving over the expert for all $\Lambda$ values. In contrast, the **ConfHAI** and **CRLogit** improve over the expert for a limited range of values of $\Lambda$, which is for a range around the true value of the sensitivity parameter $\Lambda_0$ showed in the plot, and perform noticeably worse than the expert for many $\Lambda$ values when the $\Lambda$ is mis-specified. In contrast, **CARED** shows robustness to all $\Lambda$ values, making it a safer choice, as correctly specifying $\Lambda$ is challenging [McClean et al., 2024].

## 7.2 IHDP Hidden Confounding

In this experiment, we aim to demonstrate how our method can adapt to the human expert using a semi-synthetic dataset: the IHDP hidden confounding dataset.

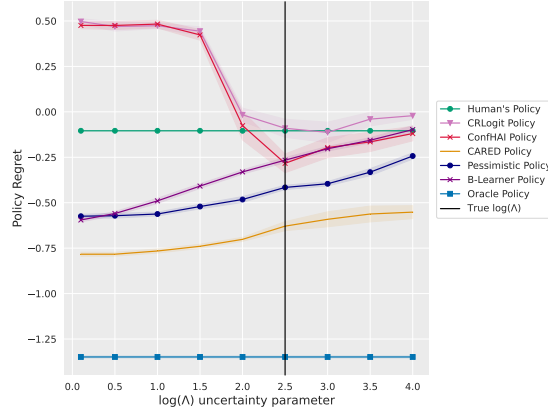

Figure 1: Synthetic Data: Policy regret, lower policy regret is better. x-axis is levels of hidden confounding according to the MSM model. The true $\Lambda_0$ is reported as a black vertical line. *Human's Policy* is the human expert's choices ($A$) as observed in the data. *CRLogit Policy* [Kallus and Zhou, 2020]: learn a policy with an IPW approach under hidden confounding, without deferral. *ConfHAI Policy* [Gao and Yin, 2023] similarly learns a policy with IPW approach under hidden confounding with deferral. *CARED*: our proposed method *Pessimistic Policy* and *B-Learner Policy* are based on CAPO bounds from the B-Learner [Oprescu et al., 2023] and are defined in Equation (2) and Equation (1), respectively. *Oracle Policy* assigns the best true treatment to each patient.

**Dataset** The hidden-confounding version of the IHDP dataset [Hill, 2011] was introduced by Jesson et al. [2021]. The Infant Health and Development Program (IHDP) dataset [Hill, 2011] is a dataset consisting of real covariates and a treatment that were collected from an RCT that targeted low-birth-weight, premature infants. The treatment was providing both intensive high-quality child care and home visits from a trained provider, while the outcomes were simulated according to the response surface B described by Hill [2011]. Jesson et al. [2021] induced hidden confounding onto the dataset by hiding the $x_9$ covariate; however, the response surface B from Hill [2011] is still used to generate the observed outcomes. In this dataset higher outcomes are better, thus we build the loss function using the appropriate set of costs for this assumption.

**Semi-Synthetic Expert** (Current Expert) We design a semi-synthetic expert, based on the original observed expert's policy, and the Oracle policy. The new semi-synthetic expert is designed as follows: when the feature $x_{17}$ -"worked during pregnancy" receives the value 1, then the new expert is identical to the Oracle policy, and otherwise, it is identical to the original expert. This way, the new expert is perfect when $x_{17}$, which happens in probability $0.59$. See details in Appendix C.2.1.

**Experiment** We conduct this experiment on the modified IHDP Hidden Confounding consisting of tuples $(X, A', Y')$, where the $A'$ is the new expert, and the $Y'$ is the outcome corresponding to the expert $A$. We generate 1000 realizations of the dataset. For each realization, we train the policy model for different values of the causal uncertainty parameter $\Lambda$ and calculate: the rate of the samples deferred to the expert, and the policy value of the learned policy. We then plot the average policy value per causal uncertainty level - the parameter $\Lambda$ in Figure 2a, and the average policy value per average deferral rate over all trials in Figure 2b. As for our method, we obtain our policy by applying Algorithm 1 with a logistic policy implemented as a single-layer MLP network.

**Results** In Figure 2a We observe that CARED consistently improves upon the expert's policy, yielding a higher average policy value across all levels of the causal uncertainty parameter, $\Lambda$. Notably, it outperforms the baseline methods for reasonable values of $\Lambda$, and achieves a policy value closest to that of the optimal Oracle Policy. As in the previous experiment, CARED demonstrates greater robustness to variations in the assumed level of confounding. For high $\Lambda$ values, CARED's performance trends toward that of the expert policy, which is expected, as larger $\Lambda$ values result in less informative and thus less reliable CAPO intervals. In contrast, ConfHAI only begins to improve when the assumed confounding level closely matches the true sensitivity parameter.

These results highlight the robustness and reliability of our policy, showing that it performs well even when the sensitivity parameter is misspecified by the data analyst. Moreover, when the sensitivity parameter reaches very high levels, CARED avoids unnecessary risk by converging towards the expert policy, reflecting its conservative response to high causal uncertainty.

In Figure 2b we compare our method only to baselines that incorporate deferral. Here, instead of regret, we assess the policies by their average policy value across all trials, and we plot the performance according to the average deferral rate over all trials. Our results show that our policy consistently outperforms the other methods at equivalent deferral rates.

The peak of the plot of the CARED Policy is when both the model and the expert work in collaboration and take advantage of the strengths of both the ML model and the expert. This happens at a deferral rate that is close to $\sim 0.6$, which is almost precisely the percentage of the cases where the expert does as well as the oracle policy according to how the expert was designed. This indicates that our model learns the strength of the expert, and knows when to defer to the expert. While ConfHAI defers only a small fraction of samples on average, which might indicate that this method does not learn the weaknesses and strengths of the expert.

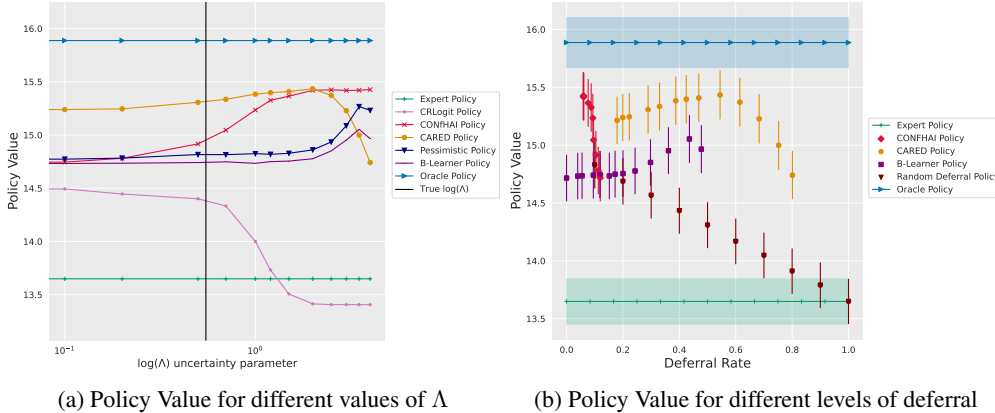

(a) Policy Value for different values of $\Lambda$    (b) Policy Value for different levels of deferral

Figure 2: IHDP Hidden Confounding: Figure 2a shows policy value for different levels of allowed hidden confounding in the data according to the MSM model Assumption 1. The x-axis represents different values of the uncertainty parameter $\Lambda$, and the true $\Lambda_0$ is reported as a black vertical line. Figure 2b shows policy value for different rates of deferral. The x-axis represents different levels of practitioner caution by varying the percentage of recommendations deferred. The methods shown here are the same as in Figure 1, in addition to *Random Deferral Policy* that defers a randomly chosen fraction of samples to the expert at each deferral rate.

## 8    Discussion

In this paper we proposed CARED: a method for learning policies from observational data where the model can recommend a treatment or defer to an expert. When learning to act from observational data which includes experts' actions, hidden confounders are by necessity factors that influenced the experts' decisions, and are thus available to them even though they are unavailable to the model. This makes our setting pertinent to the problem of *safely* learning to recommend actions based on observational data where the actions were taken by human experts, as is the case in many medical and legal settings, for example. CARED thus mitigates some of the risk of learning causal models from observational data. A further advantage is that we might not need to know the true $\Lambda$ for the system to be useful: instead we might wish to calibrate the rate of deferral instead, as that might be the more practical constraint the system faces, in terms of human labor vs. the joint system's policy value.

We showed both theoretically and by experiments in synthetic and semi-synthetic data that our method outperforms relevant baselines for this task, and can combine in a synergistic manner the expert's and machine learning model's capabilities.

**Limitations and future work** Our current method requires access to the experts actions in the observational data; it cannot accommodate directly a different expert, since we cannot know the potential outcomes corresponding to how that new expert would have acted. A further limitation is the assumption that at test time the experts would behave the same for the deferred cases as they would have before system deployment. Realistically, deploying an action recommendation system might change the experts' behavior more broadly. Accounting for this would require testing and modeling the experts behavior in such conditions. Future work will explore this more dynamic setting, taking into account the ongoing interactions and learning between the human expert and the system.

# 9 Acknowledgments

We would like to express our gratitude to our colleague, Rom Gutman, for his valuable insights and support throughout this work. We also thank Angela Zhou and Ruijiang Gao whose methods we replicated and compared in our evaluations, for their their helpful responses to our questions about implementing their methods. Additionally, we extend our appreciation to Hussein Mozannar for his generous assistance in clarifying aspects of his work, which served as the basis for our approach. We would like to thank the anonymous reviewers for useful discussion and feedback. MG and US were supported by ISF grant 2456/23.

# References

R. Adams, K. E. Henry, A. Sridharan, H. Soleimani, A. Zhan, N. Rawat, L. Johnson, D. N. Hager, S. E. Cosgrove, A. Markowski, et al. Prospective, multi-site study of patient outcomes after implementation of the trews machine learning-based early warning system for sepsis. *Nature medicine*, 28(7):1455–1460, 2022.

S. Athey and S. Wager. Policy learning with observational data. *Econometrica*, 89(1):133–161, 2021.

G. Bansal, B. Nushi, E. Kamar, E. Horvitz, and D. S. Weld. Optimizing AI for teamwork. *arXiv preprint arXiv:2004.13102*, 2020.

S. Behncke, M. Frölich, and M. Lechner. Targeting labour market programmes—results from a randomized experiment. *Swiss Journal of Economics and Statistics*, 145(3):221–268, 2009.

A. Beygelzimer and J. Langford. The offset tree for learning with partial labels. In *Proceedings of the 15th ACM SIGKDD international conference on Knowledge discovery and data mining*, pages 129–138, 2009.

J. Brooks-Gunn, F.-r. Liaw, and P. K. Klebanov. Effects of early intervention on cognitive function of low birth weight preterm infants. *The Journal of pediatrics*, 120(3):350–359, 1992.

M.-A. Charusaie, H. Mozannar, D. Sontag, and S. Samadi. Sample efficient learning of predictors that complement humans. In *International Conference on Machine Learning*, pages 2972–3005. PMLR, 2022.

A. Curth, D. Svensson, J. Weatherall, and M. van der Schaar. Really doing great at estimating cate? a critical look at ml benchmarking practices in treatment effect estimation. In *Thirty-fifth conference on neural information processing systems datasets and benchmarks track (round 2)*, 2021.

M. Dudík, D. Erhan, J. Langford, and L. Li. Doubly robust policy evaluation and optimization. 2014.

R. Gao and M. Yin. Confounding-robust policy improvement with human-ai teams. *arXiv preprint arXiv:2310.08824*, 2023.

R. Gao, M. Saar-Tsechansky, M. De-Arteaga, L. Han, M. K. Lee, and M. Lease. Human-ai collaboration with bandit feedback. In Z.-H. Zhou, editor, *Proceedings of the Thirtieth International Joint Conference on Artificial Intelligence, IJCAI-21*, pages 1722–1728. International Joint Conferences on Artificial Intelligence Organization, 8 2021.

J. L. Hill. Bayesian nonparametric modeling for causal inference. *Journal of Computational and Graphical Statistics*, 20(1):217–240, 2011.

P. W. Holland. Statistics and causal inference. *Journal of the American statistical Association*, 81(396):945–960, 1986.

A. Jesson, S. Mindermann, Y. Gal, and U. Shalit. Quantifying ignorance in individual-level causal-effect estimates under hidden confounding. In *International Conference on Machine Learning*, pages 4829–4838. PMLR, 2021.

N. Kallus. Recursive partitioning for personalization using observational data. In *International conference on machine learning*, pages 1789–1798. PMLR, 2017.

N. Kallus and A. Zhou. Minimax-optimal policy learning under unobserved confounding. *Management Science 67(5):2870-2890*, 2020.

N. Kallus, X. Mao, and A. Zhou. Interval estimation of individual-level causal effects under unobserved confounding. In *Proceedings of the 22$^{nd}$ International Conference on Aritificial Intelligence and Statistics (AISTATS) 2019*, volume 89, 2019.

T. McBrien, B. Winters, E. Zhou, and V. Eubanks. Screened & scored in the District of Columbia, November 2022. URL `https://epic.org/screened-scored-in-dc/`. [`https://epic.org/screened-scored-in-dc/`; posted November-2022].

A. McClean, Z. Branson, and E. H. Kennedy. Calibrated sensitivity models. *arXiv preprint arXiv:2405.08738*, 2024.

H. Mozannar and D. Sontag. Consistent estimators for learning to defer to an expert. In *International Conference on Machine Learning*, pages 7076–7087. PMLR, 2020.

M. Oprescu, J. Dorn, M. Ghoummaid, A. Jesson, N. Kallus, and U. Shalit. B-learner: Quasi-oracle bounds on heterogeneous causal effects under hidden confounding. In *Proceedings of the 40th International Conference on Machine Learning*, pages 26599–26618. PMLR, 2023.

P. Rajpurkar, E. Chen, O. Banerjee, and E. J. Topol. AI in health and medicine. *Nature medicine*, 28(1):31–38, 2022.

D. B. Rubin. Causal inference using potential outcomes: Design, modeling, decisions. *Journal of the American Statistical Association*, 100(469):322–331, 2005.

S. Shalev-Shwartz and S. Ben-David. *Understanding machine learning: From theory to algorithms*. Cambridge university press, 2014.

M. J. Stensrud, J. Laurendeau, and A. L. Sarvet. Optimal regimes for algorithm-assisted human decision-making. *Biometrika*, page asae016, 2024.

M. T. Stevenson and J. L. Doleac. Algorithmic risk assessment in the hands of humans. *Available at SSRN 3489440*, 2022.

A. Swaminathan and T. Joachims. Counterfactual risk minimization: Learning from logged bandit feedback. In *International Conference on Machine Learning*, pages 814–823. PMLR, 2015.

Z. Tan. A distributional approach for causal inference using propensity scores. *Journal of the American Statistical Association*, 101(476):1619–1637, 2006.

T. Yin, J.-F. Ton, R. Guo, Y. Yao, M. Liu, and Y. Liu. Fair classifiers that abstain without harm. In *The Twelfth International Conference on Learning Representations*, 2024. URL `https://openreview.net/forum?id=jvveGAbkVx`.

# A    Action Costs

In this section, we present examples that illustrate our choice of the costs. Then we prove the consistency of our costs for some base cases, that is, we show that the lowest cost corresponds to the right action for our choice for the set of costs.

Let $z = (x, a, y)$ be a sample, and assume $Y(0) > Y(1)$ w.l.o.g, that is, the right treatment is $A = 1$. $\hat{Y}^+(x,0), \hat{Y}^-(x,0), \hat{Y}^+(x,1), \hat{Y}^-(x,1)$ are the CAPOs corresponding to this sample.

We recall our proposed costs:

$$C(x, 1) = \hat{Y}^+(x, 0) - \hat{Y}^-(x, 1)$$

$$C(x, 0) = \hat{Y}^+(x, 1) - \hat{Y}^-(x, 0)$$

$$C_\perp^{cons}(x, a, y) = \begin{cases} \hat{Y}^-(x_i, 0) - y_i, & \text{if } a_i = 1 \\ \hat{Y}^-(x_i, 1) - y_i, & \text{otherwise.} \end{cases}$$

$$C_\perp^{opt}(x, a, y) = \begin{cases} \hat{Y}^+(x, 0) - y, & \text{if } a = 1 \\ \hat{Y}^+(x, 1) - y, & \text{otherwise.} \end{cases}$$

where $C_\perp^{cons}(x, a, y)$, and $C_\perp^{opt}(x, a, y)$ correspond to the *conservative* and *optimistic* deferral costs respectively.

## A.1    No overlap between CAPOs intervals

**Example** In this example, we show a case where the CAPOs intervals don't overlap, we provide a visual diagram for this example in Figure 3a, where we let $\hat{Y}^+(x, 0) = 4, \hat{Y}^-(x, 0) = 1, \hat{Y}^+(x, 1) = 9, \hat{Y}^-(x, 1) = 5$, and $Y(0) = 2, Y(1) = 6$. Then we have that:

$$C(x, 1) = -1$$
$$C(x, 0) = 8$$

As for the deferral cost, there are two possible cases: if the expert is right, meaning $a = 1, y = Y(1)$, the deferral costs are:

$$C_\perp^{cons}(x, a, y) = -5$$
$$C_\perp^{opt}(x, a, y) = -2$$

for both alternatives, we have the lowest costs corresponding to the deferral cost, which means that we will guide the model to choose the expert's decision, which is in this case, the right treatment.

On the other hand, when the expert is wrong, i.e. $a = 0, y = Y(0)$, the deferral costs are:

$$C_\perp^{cons}(x, a, y) = 3$$
$$C_\perp^{opt}(x, a, y) = 7$$

Where for both alternatives, the lowest cost is the cost of the treatment $a = 1$, which is the right treatment.

**Proof of the general case** We show a visualization for the general case where the CAPOs intervals don't overlap in Figure 3b. In the general case we have that:

$$C(x, 1) = -d$$
$$C(x, 0) = \ell_0 + d + \ell_1$$

As for the deferral cost, when expert is right, meaning $a = 1, y = Y(1)$, the deferral costs are:

$$C_\perp^{cons}(x, a, y) = -\left(\ell_0 + d + \ell_1^-\right)$$
$$C_\perp^{opt}(x, a, y) = -\left(d + \ell_1^-\right)$$

for both alternatives, we have the lowest costs corresponding to the deferral cost, which means that we will guide the model to choose the expert's decision, which is in this case, the right treatment.

On the other hand, when the expert is wrong, i.e. $a = 0, y = Y(0)$, the deferral costs are:

$$C_\perp^{cons}(x, a, y) = \ell_0^+ + d$$
$$C_\perp^{opt}(x, a, y) = \ell_1 + d + \ell_0^+$$

Where for both alternatives, the lowest cost is the cost of the treatment $a = 1$, which is the right treatment.

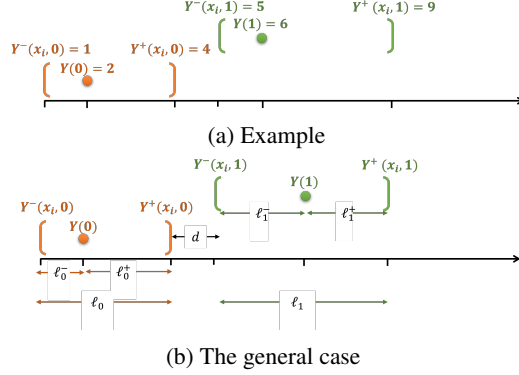

(a) Example

(b) The general case

Figure 3: No overlap between CAPOs intervals visualization

## A.2 CAPOs intervals overlap

We now focus on the cases where the CAPOs intervals overlap. In Figure 4 we present two examples for cases where the CAPOs intervals overlap and show how this affects the costs.

**Example 1.** In the example presented in Figure 4a, where we let $\hat{Y}^+(x, 0) = 5, \hat{Y}^-(x, 0) = 1, \hat{Y}^+(x, 1) = 9, \hat{Y}^-(x, 1) = 4$, and $Y(0) = 2, Y(1) = 6$. Then we have that:

$$C(x, 1) = 1$$
$$C(x, 0) = 8$$

As for the deferral cost, there are two possible cases: if the expert is right, meaning $a = 1, y = Y(1)$, the deferral costs are:

$$C_\perp^{cons}(x, a, y) = -5$$
$$C_\perp^{opt}(x, a, y) = -1$$

for both alternatives, we have the lowest costs corresponding to the deferral cost, which means that we will guide the model to choose the expert's decision, which is in this case, the right treatment.

On the other hand, when the expert is wrong, i.e. $a_i = 0, y_i = Y(0)$, the deferral costs are:

$$C_\perp^{cons}(x, a, y) = 2$$
$$C_\perp^{opt}(x, a, y) = 7$$

Where for both alternatives, the lowest cost is the cost of the treatment $a = 1$, which is the right treatment.

**Example 2.** In the example presented in Figure 4b, we show an interesting case, where although $Y(1) > Y(0)$, the value of $Y(1)$ lies into the intersection of the two intervals. We let $\hat{Y}^+(x, 0) = 5, \hat{Y}^-(x, 0) = 1, \hat{Y}^+(x, 1) = 9, \hat{Y}^-(x, 1) = 4$, and $Y(0) = 2, Y(1) = 4.5$. Then we have that:

$$C(x, 1) = 1$$
$$C(x, 0) = 8$$

As for the deferral cost, there are two possible cases: if the expert is right, meaning $a = 1, y = Y(1)$, the deferral costs are:

$$C_\perp^{cons}(x, a, y) = -3.5$$
$$C_\perp^{opt}(x, a, y) = 0.5$$

for both alternatives, we have the lowest costs corresponding to the deferral cost, which means that we will guide the model to choose the expert's decision, which is in this case, the right treatment.

On the other hand, when the expert is wrong, i.e. $a_i = 0, y_i = Y(0)$, the deferral costs are:

$$C_\perp^{cons}(x, a, y) = 2$$
$$C_\perp^{opt}(x, a, y) = 7$$

Where for both alternatives, the lowest cost is the cost of the treatment $a = 1$, which is the right treatment.

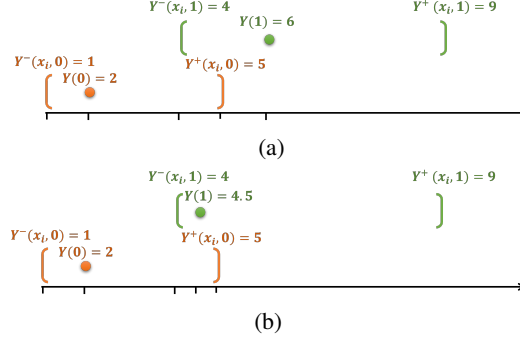

(a)

(b)

Figure 4: CAPOs intervals overlap visualization

## A.3  A Comparative Analysis of the Conservative and Optimistic Costs

We provide an analysis that highlights the scenarios where each of the conservative and optimistic approaches proves superior to the other.

For a sample $(x, a, y, Y(0), Y(1)) \sim P_{\text{full}}$, where w.l.o.g it holds that $Y(1) > Y(0)$, that is, the right treatment for this sample is 1.

We distinguish between two main cases:

**The expert is right**:
When the expert is right, i.e. $\pi_{\exp}(x) = 1$, and $Y(\pi_{\exp}(x_i)) = Y(1)$, then, a policy with this set of costs will predict the wrong treatment $\tilde{\pi}(x) = 0$ when the following condition holds:

$$C(x, 0) < \min\{C(x, 1), C_\perp(x, a = 1, y = Y(1))\} \tag{6}$$

For *the conservative approach*, this holds when:

$$\hat{Y}^+(x, 1) - \hat{Y}^-(x, 0) < \hat{Y}^-(x, 0) - Y(1) \tag{7}$$

As for the *the optimistic approach*, this holds when:

$$\hat{Y}^+(x, 1) - \hat{Y}^-(x, 0) < \hat{Y}^+(x, 0) - Y(1) \tag{8}$$

When Equation (7) holds and Equation (8) does not hold, these are the cases where the *optimistic* approach outperforms the *conservative* approach, i.e.:

$$\hat{Y}^+(x, 0) + \hat{Y}^-(x, 0) < \hat{Y}^+(x, 0) + Y(1) < 2\hat{Y}^-(x, 0) \tag{9}$$

We note that Equation (14) never holds, and thus when the expert is right, the optimistic approach cannot outperform its conservative counterpart.

On the other hand, when Equation (8) holds and Equation (7) does not hold, these are the cases where the *conservative* approach outperforms the *optimistic* approach, i.e.:

$$2\hat{Y}^-(x, 0) < \hat{Y}^+(x, 1) + Y(1) < \hat{Y}^-(x, 0) + \hat{Y}^+(x, 0) \tag{10}$$

**The expert is wrong**
When the expert is wrong, i.e. $\pi_{\exp}(x) = 0$, and $Y(\pi_{\exp}(x_i)) = Y(0)$, then, a policy with this set of costs will predict the wrong treatment $\tilde{\pi}(x) = 0$ when the following condition holds:

$$\min\{C(x, 0), C_\perp(x, a = 0, y = Y(0))\} < C(x, 1) \tag{11}$$

For *the conservative approach*, this holds when:

$$\hat{Y}^-(x, 1) - Y(0) < \hat{Y}^+(x, 0) - \hat{Y}^-(x, 1) \tag{12}$$

As for the *the optimistic approach*, this holds when:

$$\hat{Y}^+(x,1) - Y(0) < \hat{Y}^+(x,0) - \hat{Y}^-(x,1) \tag{13}$$

When Equation (12) holds and Equation (13) does not hold, these are the cases where the *optimistic* approach outperforms the *conservative* approach, i.e.:

$$2\hat{Y}^-(x,1) < \hat{Y}^+(x,0) + Y(0) < \hat{Y}^+(x,1) + \hat{Y}^-(x,1) \tag{14}$$

On the other hand, when Equation (13) holds and Equation (12) does not hold, these are the cases where the *conservative* approach outperforms the *optimistic* approach, i.e.:

$$\hat{Y}^-(x,1) + \hat{Y}^+(x,1) < \hat{Y}^+(x,0) + Y(0) < 2\hat{Y}^-(x,1) \tag{15}$$

We note that Equation (14) never holds, and thus when the expert is wrong, the conservative approach cannot outperform its optimistic counterpart.

# B  Proof of Main Theorems

In this section, we provide the proof of our main theorems.

### B.0.1  Consistency

*Proof of Corollary 1.* we can apply Proposition 1 from Mozannar and Sontag [2020] directly to our setting yielding the statement of our Corollary. □

Note that we used a slightly different statement of Proposition 1 from Mozannar and Sontag [2020] from readability considerations and consistency with our setup and problem formulation.

The formulation of the original statement using our notation is as follows:

**Corollary 2.** $L_{CE}$ *is convex in* $\pi$ *and is a consistent loss function for* $L$: *Let* $\tilde{\pi} = \arg\inf_\pi \mathbb{E}[L_{CE}(\pi, z; \hat{Q})]$, *then* $\arg\max_{i \in \{0,1,\perp\}} \tilde{\pi}_i = \arg\min_{i \in \{0,1,\perp\}} \mathbb{E}[c(i)|Z = z]$.

with $c(0) = C(x,0)$, $c(1) = C(x,1)$, and $c(\perp) = C_\perp(x,a,y)$ defined in Section 5.

We recall that the surrogate loss $L_{CE}$[4] is defined for policies $\pi_i : \mathcal{X} \to \mathbb{R}$ be the raw output of the policy $\pi$ corresponding to a class $i \in \{0,1,\perp\}$, such that $\pi(x) = \arg\max_{i \in \{0,1,\perp\}} \pi_i(x)$. Therefore, this holds especially for $\tilde{\pi} = \arg\inf_\pi \mathbb{E}[L_{CE}(\pi, z; \hat{Q})]$, i.e.

$$\tilde{\pi} = \arg\max_{i \in \{0,1,\perp\}} \tilde{\pi}_i \tag{16}$$

As for optimizing the original loss function $L(\pi)$[3], the optimization problem is given by,

$$\dot{\pi} = \arg\inf_{\pi'} \mathbb{E}[L(\pi')] \tag{17}$$

according to Mozannar and Sontag [2020], this problem can be solved as a cost-sensitive problem with the costs $c(0) = C(x,0)$, $c(1) = C(x,1)$, and $c(\perp) = C_\perp(x,a,y)$ for each sample $(x,a,y)$, i.e., the solution of this optimal optimization problem satisfies the following for each sample $z$: $\dot{\pi} = \arg\min_{i \in \{0,1,\perp\}} \mathbb{E}[c(i)|Z = z]$ Thus, our adaptation of the results is equivalent to the original results from Mozannar and Sontag [2020].

## B.1  Costs Are Coherent

In this section, we prove the coherency of our costs as defined in Section 6.2, and show the improvement of the CARED policy over policies that depend solely on either a human expert or a machine.

*Proof of Theorem 1.* For a sample $(x, a, y, Y(0), Y(1)) \sim P_{\text{full}}$, where w.l.o.g it holds that $Y(1) > Y(0)$, that is, the right treatment for this sample is 1. We assume Definition 1 holds with probability $1 - \delta$. Then we prove this theorem in two steps: Comparison against the human expert policy ($\pi_{exp}$), and comparison against the bounds policy($\pi_{bounds}$).

**Comparison Against the Expert**:

We distinguish between two main cases:

**The expert is right**:

When the expert is right, i.e. $\pi_{\exp}(x) = 1$, and $Y(\pi_{\exp}(x_i)) = Y(1)$, then our policy will predict the wrong treatment $\tilde{\pi}(x) = 0$ when the following condition holds:

$$C(x,0) < \min\{C(x,1), C_\perp(x, a = 1, y = Y(1))\} \tag{18}$$

That is:

$$\hat{Y}^+(x,1) - \hat{Y}^-(x,0) < \min\{\hat{Y}^+(x,0) - \hat{Y}^-(x,1), \hat{Y}^-(x,0) - Y(1)\} \tag{19}$$

In simple words, we recall that our method assigns the treatment with the minimal cost among all other treatments. Thus, when the cost of the wrong treatment is the minimal cost among the other costs, our method will make mistakes.

We note that it holds that:

$$\hat{Y}^-(x,0) - Y(1) \underset{\hat{Y}^-(x,1)\leq Y(1)}{\leq} \hat{Y}^-(x,0) - \hat{Y}^-(x,1) \underset{\hat{Y}^+(x,0)\geq \hat{Y}^-(x,0)}{\leq} \hat{Y}^+(x,0) - \hat{Y}^-(x,1)$$

$$\Rightarrow \min\{\hat{Y}^+(x,0) - \hat{Y}^-(x,1), \hat{Y}^-(x,0) - Y(1)\} = \hat{Y}^-(x,0) - Y(1)$$

Thus, $\tilde{\pi}(x)$ is wrong when:

$$\hat{Y}^+(x,1) - \hat{Y}^-(x,0) < \hat{Y}^-(x,0) - Y(1) \tag{20}$$

We note that the condition in 20 never holds when CAPO bounds are valid, and therefore, with probability $1-\delta$, it holds that $\tilde{\pi}(x) = 1$, and $Y(\tilde{\pi} = (x)) = Y(1)$, That is, $Y(\tilde{\pi}(x)) = Y(\pi_{\exp}(x))$

**The expert is wrong**:

When the expert is wrong, i.e. $\pi_{\exp}(x) = 0$, and $Y(\pi_{\exp}(x)) = Y(0)$, then, from the same considerations above, our policy is wrong when:

$$\min\{C(x,0), C_\perp(x, a=0, y=Y(0))\} < C(x,1) \tag{21}$$

That is:

$$\min\{\hat{Y}^+(x,1) - \hat{Y}^-(x,0), \hat{Y}^-(x,1) - Y(0)\} < \hat{Y}^+(x,0) - \hat{Y}^-(x,1) \tag{22}$$

We note that it holds that:

$$\hat{Y}^-(x,1) - Y(0) \underset{\hat{Y}^-(x,0)\leq Y(0)}{\leq} \hat{Y}^-(x,1) - \hat{Y}^-(x,0) \underset{\hat{Y}^+(x,1)\geq \hat{Y}^-(x,1)}{\leq} \hat{Y}^+(x,1) - \hat{Y}^-(x,0)$$

$$\Rightarrow \min\{\hat{Y}^+(x,1) - \hat{Y}^-(x,0), \hat{Y}^-(x,1) - Y(0)\} = \hat{Y}^-(x,1) - Y(0)$$

Thus, $\tilde{\pi}(x)$ is wrong when:

$$\hat{Y}^-(x,1) - Y(0) < \hat{Y}^+(x,0) - \hat{Y}^-(x,1) \tag{23}$$

When this happens, we have that $\tilde{\pi}(x) = 0$, and $Y(\tilde{\pi}(x)) = Y(0)$, that is, $Y(\tilde{\pi}(x_i)) = Y(\pi_{\exp}(x_i))$, otherwise, when this condition does not hold, we have that $Y(\tilde{\pi}(x)) > Y(\pi_{\exp(x)})$

**Comparison Against the Bounds Policy**

We analyze the cases where each policy makes mistakes. We note that $\tilde{\pi}$ is always right when CAPO intervals don't overlap as proved in Appendix A. This holds for the $\pi_{bounds}$ policy as well, as this implies directly from its definition. On the other hand, when the CAPO bounds overlap, both policies can make mistakes

Thus, we distinguish between two cases:

**CAPO bounds don't overlap** In this case, both policies recommend the right action, thus it holds that, $Y(\tilde{\pi}(x)) = Y(\pi_{bounds}(x)) = Y(1)$.

**CAPO Bounds don't Overlap** In this case, $\pi_{bounds}$ defer the decision to the expert. As for $\tilde{\pi}$, as shown above, the policy $\tilde{\pi}$ is wrong when the expert is wrong and the condition 23 holds. Therefore, if condition 23 holds, and the expert is wrong, then both policies will go wrong, and thus we have that $Y(\tilde{\pi}(x)) = Y(\pi_{bounds}(x)) = Y(0)$.

On the other hand, if CAPO intervals overlap, but Condition 23 doesn't hold, then $\tilde{\pi}$ give the right recommendation with $Y(\tilde{\pi}(x)) = Y(1)$, but $\pi_{bounds}$ still give the wrong recommendation with $Y(\pi_{bounds}(x)) = Y(0)$, that is $Y(\tilde{\pi}(x)) > Y(\pi_{bounds}(x))$.

$$\square$$

## B.2 Generalization Bound

We now prove Theorem 2.

**Corollary 3** (Bounded log-Softmax). *Let $\pi \in \Pi$ be a learner that satisfies Assumption 3. Then, the term $\frac{\exp(\pi_j(x))}{\sum_{k \in \{0,1,\perp\}} \exp(\pi_k(x))}$ is bounded.*

*Proof of Corollary 3.* $\pi$ is a policy satisfying Assumption 3, that is for each $j \in \{0,1,\perp\}$ we have that $|\pi_j(\cdot)| \leq C_\pi$, where $K$ is the number of classes. That is, it holds that

$$-C_\pi \leq \pi_j(\cdot) \leq C_\pi$$
$$\Leftrightarrow \exp(-C_\pi) \leq \exp(\pi_j) \leq \exp(C_\pi)$$

As a result, we have,

$$\frac{\exp(\pi_j(x))}{\sum_{k \in \{0,1,\perp\}} \exp(\pi_k(x))} \leq \frac{\exp(C_\pi)}{\sum_{k \in \{0,1,\perp\}} \exp(-C_\pi)}$$
$$\leq \frac{\exp(C_\pi)}{3 \cdot \exp(-C_\pi)}$$
$$\leq \frac{1}{3} \cdot \exp(2 \cdot C_\pi)$$

Similarly, we can show that

$$\frac{\exp(\pi_j(x))}{\sum_{k \in \{0,1,\perp\}} \exp(\pi_k(x))} \geq \frac{1}{3} \cdot \exp(-2 \cdot C_\pi)$$

$\square$

**Lemma 1** (Bounded loss function). *Let $\Pi$ be a class of policy learners satisfying Assumption 3 with a constant $C_\pi$, and suppose Assumption 4 holds with a constant $C_Y$. Then $\left| L_{CE}(\pi, z; \hat{Q}) \right| \leq 8 \cdot C_\pi \cdot C_Y$.*

*Proof of Lemma 1.* Let $\Pi$ be a class of policy learners satisfying Assumption 3, then the loss function $L_{CE}(\pi, z_i; \hat{Q}(x_i))$ is bounded:

$$|L_{CE}(\pi, z; Q)| = \left| \sum_{j \in \{0,1,\perp\}} -w^j(z, Q(x)) \log\left(\frac{\exp(\pi_j(x))}{\sum_{k \in \{0,1,\perp\}} \exp(\pi_k(x))}\right) \right|$$

$$= \left| \sum_{j \in \{0,1,\perp\}} w^j(z, Q(x)) \log\left(\frac{\exp(\pi_j(x))}{\sum_{k \in \{0,1,\perp\}} \exp(\pi_k(x))}\right) \right|$$

$$\underset{(a)}{\leq} \left| \sum_{j \in \{0,1,\perp\}} w^j(z, Q(x)) \log\left(\frac{1}{3} \cdot \exp(2 \cdot C_\pi)\right) \right|$$

$$\leq \left| \sum_{j \in \{0,1,\perp\}} w^j(z, Q(x)) \cdot 2 \cdot C_\pi \right|$$

$$\leq 2 \cdot C_\pi \cdot \sum_{j \in \{0,1,\perp\}} \left| w^j(z, Q(x)) \right|$$

$$\underset{(b)}{\leq} 2 \cdot C_\pi \cdot |4 \cdot C_Y| = 8 \cdot C_\pi \cdot C_Y$$

Where step $(a)$ follows from Corollary 3, and $(b)$ follows from the boundedness of $Y$ in Assumption 4. $\square$

**Lemma 2** (L2 Consistency of the $\max(\cdot, \cdot)$ Estimator). *Let For estimators $\hat{A}_n, \hat{B}_n$ of $A, B$ based on $n$ samples respectively, where there exists a function $f(n)$ such that:*

$$\|\hat{A}_n - A\| \lesssim O_p(f(n))$$
$$\|\hat{B}_n - B\| \lesssim O_p(f(n))$$

*Then,*

$$\|\max(\hat{A}_n, \hat{B}_n) - \max(A, B)\| \lesssim O_p(f(n))$$

*Proof.* We prove this Lemma in two steps: first we show convergence in probability of $\max(\hat{A}_n, \hat{B}_n)$ to $\max(A, B)$. Then we show the convergence rate.

**Convergence** $\hat{A}_n$ and $\hat{B}_n$ converge in probability to $A$ and $B$ respectively as $n$ tends to infinity, that is:

$$\hat{A}_n \xrightarrow[n\to\infty]{P} A$$

$$\hat{B}_n \xrightarrow[n\to\infty]{P} B$$

Then,

$$\max(\hat{A}_n, \hat{B}_n) = \frac{1}{2}\left(\hat{A}_n + \hat{B}_n + \left|\hat{A}_n - \hat{B}_n\right|\right) \xrightarrow[n\to\infty]{P} \frac{1}{2}(A + B + |A - B|) = \max(A, B)$$

**Convergence Rate** The convergence rate of $\max(\hat{A}_n, \hat{B}_n)$ is determined by the slower convergence rate of $\hat{A}_n$ and $\hat{B}_n$, which is $\max(f(n), f(n)) = f(n)$ $\qquad\square$

**Corollary 4** (Bounded weights)**.** *The weights $w^j(z, \hat{Q}(x))$ for $j \in \{0, 1, \perp\}$ of the weighted surrogate loss function 4 based on the costs we define in Section 5.2 satisfy:*

*(i)* $\left|w^j(z, Q(x))\right| \leq 4 \cdot C_Y$
*(ii)* $\|w^j(z, Q(x)) - w^j(z, \hat{Q}(x))\| \lesssim 4 \cdot O_p\left(n^{-1/(2+r)}\right)$

*Proof.* We recall that for $j \in \{0, 1, \perp\}$, a cost $c(j)$ is of the form $c(j) = A - B$, where $A, B \in \{\hat{Y}^+(x, 0), \hat{Y}^-(x, 0), \hat{Y}^+(x, 1)), \hat{Y}^-(x, 1), y_i\}$.

As for $\max_{k \in \{0,1,\perp\}} c(k)$ is also of the form $\max_{k \in [K+1]} c(k) = C - D$, where $C, D \in \{\hat{Y}^+(x, 0), \hat{Y}^-(x, 0), \hat{Y}^+(x, 1)), \hat{Y}^-(x, 1), y_i\}$. Then,

(i) $\left|w^j(z, Q(x))\right| = |(A - B) - (C - D)| \leq |A| + |B| + |C| + |D| \underset{(a)}{\leq} 4 \cdot C_Y.$

where $(a)$ follows from the boundedness of the outcomes (Assumption 4 with a constant $C_Y$).

(ii) For $A, B, C, D \in \{Y^+(x, 0), Y^-(x, 0), Y^+(x, 1)), Y^-(x, 1), y_i\}$, we denote $\hat{A}, \hat{B}, \hat{C}, \hat{D}$ to be their estimators, respectively. Then,

$$\begin{aligned}\|w^j(z, Q(x)) - w^j(z, \hat{Q}(x))\| &= \|(A - \hat{A}) - (B - \hat{B}) - (C - \hat{C}) + (D - \hat{D})\| \\ &\leq \|A - \hat{A}\| + \|B - \hat{B}\| + \|C - \hat{C}\| + \|D - \hat{D}\| \\ &\underset{(a)}{\lesssim} 4 \cdot O_p\left(n^{-1/(2+r)}\right)\end{aligned}$$

where $(a)$ follows from Assumption 5, and Lemma 2. $\qquad\square$

*Proof of Theorem 2.* Let $\pi \in \Pi$ be a policy where the policy class $\Pi$ satisfies Assumption 2, for a policy $\pi \in \Pi$, and $Q, \hat{Q}$ the CAPOs bounds and the estimated CAPOs bounds respectively. Given our loss function $L_{CE}(\pi, z; Q)$ which satisfies Assumption 3, and Lemma 1. We are interested in bounding the term

$$L_D(\pi; Q) - L_S(\pi; \hat{Q}) = \underbrace{\left(L_D(\pi; Q) - L_D(\pi; \hat{Q})\right)}_{A} + \underbrace{\left(L_D(\pi; \hat{Q}) - L_S(\pi; \hat{Q})\right)}_{B} \qquad (24)$$

We get the upper bound for *Term B* by applying Theorem 26.5 from Shalev-Shwartz and Ben-David [2014], that is, with probability at least $1 - \delta$ we have:

$$L_D(\pi; \hat{Q}) - L_S(\pi; \hat{Q}) \leq 2\mathcal{R}_n(\Pi) + 24 \cdot C_\pi \cdot C_Y \sqrt{\frac{2\ln(4/\delta)}{n}}$$

where $\mathcal{R}_n(\Pi)$ is the Rademacher Complexity of the policy class $\Pi$ we have from Assumption 2, $C_\pi$, and $C_Y$ are the constants we have from Assumption 3, and Assumption 4 respectively.

We now show the upper bound for *Term A*:

$$L_D(\pi; Q) - L_D(\pi; \hat{Q}) = \mathbb{E}_{z=(x,a,y)\sim p(z)}\left[L_{CE}(\pi, z; Q(x))\right] - \mathbb{E}_{z=(x,a,y)\sim p(z)}\left[L_{CE}(\pi, z; \hat{Q}(x))\right]$$

$$= \mathbb{E}_{z=(x,a,y)\sim p(z)}\left[\sum_{j\in\{0,1,\perp\}} w^j(z, Q(x)) \cdot -\log\left(\frac{\exp(\pi_j(x))}{\sum_{k\in\{0,1,\perp\}}\exp(\pi_k(x))}\right)\right]$$

$$- \mathbb{E}_{z=(x,a,y)\sim p(z)}\left[\sum_{j\in\{0,1,\perp\}} w^j(z, \hat{Q}(x)) \cdot -\log\left(\frac{\exp(\pi_j(x))}{\sum_{k\in\{0,1,\perp\}}\exp(\pi_k(x))}\right)\right]$$

$$= \mathbb{E}_{z=(x,a,y)\sim p(z)}\left[\sum_{j\in\{0,1,\perp\}} \left(w^j(z, Q(x)) - w^j(z, \hat{Q}(x))\right) \cdot\right.$$

$$\left. -\log\left(\frac{\exp(\pi_j(x))}{\sum_{k\in\{0,1,\perp\}}\exp(\pi_k(x))}\right)\right]$$

$$\underset{(a)}{\leq} \mathbb{E}_{z=(x,a,y)\sim p(z)}\left[\sum_{j\in\{0,1,\perp\}} \left(w^j(z, Q(x)) - w^j(z, \hat{Q}(x))\right) \cdot (2 \cdot C_\pi)\right]$$

$$\leq 2 \cdot C_\pi \cdot \mathbb{E}_{z=(x,a,y)\sim p(z)}\left[\sum_{j\in\{0,1,\perp\}} \left(w^j(z, Q(x)) - w^j(z, \hat{Q}(x))\right)\right]$$

$$\underset{(b)}{\leq} 2 \cdot C_\pi \cdot \sum_{j\in\{0,1,\perp\}} \mathbb{E}_{z=(x,a,y)\sim p(z)}\left[w^j(z, Q(x)) - w^j(z, \hat{Q}(x))\right]$$

$$\underset{(c)}{\leq} 2 \cdot C_\pi \cdot \sum_{j\in\{0,1,\perp\}} \left|\mathbb{E}_{z=(x,a,y)\sim p(z)}\left[w^j(z, Q(x)) - w^j(z, \hat{Q}(x))\right]\right|$$

$$\underset{(d)}{=} 2 \cdot C_\pi \cdot \sum_{j\in\{0,1,\perp\}} \sqrt{\mathbb{E}_{z=(x,a,y)\sim p(z)}\left[w^j(z, Q(x)) - w^j(z, \hat{Q}(x))\right]^2}$$

$$\underset{(e)}{\leq} 2 \cdot C_\pi \cdot \sum_{j\in\{0,1,\perp\}} \sqrt{\mathbb{E}_{z=(x,a,y)\sim p(z)}\left[\left(w^j(z, Q(x)) - w^j(z, \hat{Q}(x))\right)^2\right]}$$

$$\underset{(f)}{\lesssim} 2 \cdot C_\pi \cdot 3 \cdot 4 \cdot O_p\left(n^{-1/(2+r)}\right)$$

$$\underset{(g)}{\leq} 24 \cdot C_\pi \cdot C \cdot \left(n^{-1/(2+r)}\right)$$

where:

$(a)$ : *Corollary* 3.

$(b)$ : Linearity of Expectation.

$(c)$ : $\mathbb{E}[X] \leq |\mathbb{E}[X]|$.

$(d)$ : $|\mathbb{E}[X]| = \sqrt{\mathbb{E}[X]^2}$.

$(e)$ : Jensen's Inequality: $\mathbb{E}[X]^2 \leq \mathbb{E}[X^2]$.

$(f)$ : *Assumption* 5.

$(g)$ : *Corollary* 4, and Definition of $O_p(\cdot)$ with constant $C > 0$.

Putting it all together, we have:

$$L_D(\pi; Q) - L_S(\pi; \hat{Q}) = \left(L_D(\pi; Q) - L_D(\pi; \hat{Q})\right) + \left(L_D(\pi; \hat{Q}) - L_S(\pi; \hat{Q})\right)$$

$$\leq 2\mathcal{R}_n(\Pi) + 24 \cdot C_\pi \cdot \left(C_Y\sqrt{\frac{2\ln(4/\delta)}{n}} + C \cdot \left(n^{-1/(2+r)}\right)\right)$$

□

# C Additional Experimental Detail

The experiments in this paper were conducted using a PowerEdge R750XA Server with 2 CPUs and 4 NVIDIA A40 GPUs. Here we provide all the details required to replicate the paper results. In addition, we provide replication code at `https://github.com/marahgh/CARED`.

## C.1 Synthetic Data

In this experiment, We used the synthetic dataset from Gao and Yin [2023]The CAPO bounds were estimated using `XGBRegressor` from `xgboost` as the base learners for the B-Learner [Oprescu et al., 2023] estimator, and a `LogisticRegression` from `scikit-learn` as the propensity score estimator. We show in Table 1 the hyper-parameter choices for each model. As for the policy model, we use a logistic regression model implemented as a one-layer MLP using several functions from `pytorch`, The policy model was implemented using `pytorch_lightning` model as a wrapper model that can receive any policy model as the base model for learning the policy. Moreover, the hyper-parameters we use for the policy model are as follows: `learning_rate`$= 0.001$, `optimizer`$= Adam$, `patience`$= 3$, and `max_epochs`$= 100$.

We replicate the experiment from Gao and Yin [2023] where they generate 10 instances of the synthetic dataset according to the Data Generation Process they mention in their paper, where lower outcomes are assumed to be better in this dataset. For each instance, they vary the level of allowed hidden confounding and compare the methods by their regret related to the baseline no-treat policy, i.e. $\pi_0(x) = 0$.

Table 1: Hyper-parameters for model choices in the synthetic data experiment

| Model | Hyper-parameter | Value |
|---|---|---|
| XGBRegressor (xgboost) | learning_rate | 0.1 |
| | min_child_weight | 3 |
| | max_depth | 5 |
| | n_estimators | 200 |
| Logistic Regression (scikit-learn) | C | 1 |
| | penalty | elasticnet |
| | solver | saga |
| | l1_ratio | 0.7 |
| | max_iter | 100 |

## C.2 IHDP Dataset

In this experiment, we use the hidden confounding version of IHDP [Hill, 2011] which was introduced by Jesson et al. [2021]. The CAPO bounds were estimated using `XGBRegressor` from `xgboost` as the base learners for the B-Learner [Oprescu et al., 2023] estimator, and a `LogisticRegression` from `scikit-learn` as the propensity score estimator. We show in Table 2 see the hyper-parameter choices for each model. As for the policy model, we use a logistic regression model implemented as a single-layer MLP using several functions from `pytorch`, The policy model was implemented using `pytorch_lightning` model as a wrapper model that can receive any policy model as the base model for learning the policy. Moreover, the hyper-parameters were tuned for each uncertainty level $\Lambda$ using the `ray.tune` over the search space of the hyper-parameters: The search spaces used is `learning_rate` $\in [1e-4, 0.1]$ ,`optimizer` $\in [SGD, Adam, AdamW]$, `weight_decay`$\in [1e-10, 1e-3]$, `patience` $\in [5, 20]$, and `max_epochs`$\in [30, 50]$.

As for Gao and Yin [2023]'s method, we train a logistic policy with the hyper-parameters: `learning_rate` $= 0.01$, `batch_size` $= 32$, `max_epochs` $= 100$. The hyper-parameters were tuned using a grid search over a set of possible values for each parameter, and the set with the most reasonable loss was chosen. In addition, we set $C(X) = 0$ which represents the additional cost of deferral used in Gao and Yin [2023]'s objective for cases where the outcome of deferral is $Y + C(X)$. In this experiment, the outcome of deferral to the expert is solely based on the result of the treatment prescribed by the expert, which is $Y$.

We generate 1111 instances of the Hidden Confounding IHDP [Jesson et al., 2021], each consists of training ($n = 470$), validation ($n = 202$), and test ($n = 75$) subsets, where for each instance the seed runs over $0, 1, ...,$ which is the number of the trial/instance.

We filter the 1111 instances by the value $\sigma_{CATE} = \sqrt{Var(CATE_{test}(X))}$ by excluding trials with high $\sigma_{CATE}$, as instances with high $\sigma_{CATE}$ are unrealistic and does not match the results of the original study [Brooks-Gunn et al., 1992], as explained in Curth et al. [2021]. In our case, based on the histogram of $\sigma_{CATE}$ for the 1111 instances of the dataset, we exclude those with $\sigma_{CATE} > 15$ and remain with 1000 instances.

Table 2: Hyper-parameters for model choices in IHDP experiment

| Model | Hyper-parameter | Value |
|---|---|---|
| XGBRegressor (xgboost) | learning_rate | 0.05 |
| | min_child_weight | 5 |
| | max_depth | 3 |
| | n_estimators | 500 |
| Logistic Regression (scikit-learn) | C | 1 |
| | penalty | elasticnet |
| | solver | saga |
| | l1_ratio | 0.7 |
| | max_iter | 10000 |

### C.2.1 Semi-Synthetic Expert

In this experiment, we design a semi-synthetic expert, based on the original observed expert's policy, and the outcomes observed under this policy. The actions and outcomes under this policy are denoted $A$ and $Y$, respectively, as they are (trivially) exactly the observed actions and outcomes. Let $A'$ and $Y'$ denote the new actions by the new expert's policy and the observed outcomes under this new policy respectively. We recall that the actions and the outcomes generated by the Oracle policy are denoted by $A^*$, and $Y^*$ respectively. Then, for $i \in [n]$, the new expert policy is given by:

$$A'_i = \begin{cases} A^*_i, & \text{if } x_i^{17} = 1 \\ A_i, & \text{otherwise} \end{cases}$$

$$Y'_i = \begin{cases} Y(0)_i, & \text{if } A'_i = 0 \\ Y(1)_i, & \text{otherwise.} \end{cases}$$

The new expert $A'$ is defined based on the covariate $x_{17}$ - "worked during pregnancy", which is a binary covariate that indicates whether the mother worked during the pregnancy or not. This covariate receives a value of 1 with a probability of 0.59. We build the new expert $A'$ to be equal to the oracle policy when $x_{17} = 1$, and equal to the original policy $A$ otherwise. Thus this expert is perfect when $x_{17} = 1$. This is an aspect of the expert that ideally we would want the model to learn. The choice of covariate $x_{17}$ was according to the covariate table shown in Jesson et al. [2021], where they show an analysis of the relationship between each covariate and both the treatment and the outcome. We choose the covariate $x_{17}$ as it is one of the features that is correlated with both the treatment and the outcome, and thus, an important feature that we are interested in testing our model on its ability to learn it.

